# MiSO: Optimizing brain stimulation to create neural population activity states

**Yuki Minai**[1,2,5], **Joana Soldado-Magraner**[1,4,5], **Matthew A. Smith**[1,3,5*], **Byron M. Yu**[1,3,4,5*]

[1]Neuroscience Institute, Carnegie Mellon University
[2]Machine Learning Department, Carnegie Mellon University
[3]Department of Biomedical Engineering, Carnegie Mellon University
[4]Department of Electrical and Computer Engineering, Carnegie Mellon University
[5]Center for the Neural Basis of Cognition
{yminai,jsoldado,msmith,byronyu}@andrew.cmu.edu
*Denotes equal contribution.

## Abstract

Brain stimulation has the potential to create desired neural population activity states. However, it is challenging to search the large space of stimulation parameters, for example, selecting which subset of electrodes to be used for stimulation. In this scenario, creating a model that maps the configuration of stimulation parameters to the brain's response can be beneficial. Training such an expansive model usually requires more stimulation-response samples than can be collected in a given experimental session. Furthermore, changes in the properties of the recorded activity over time can make it challenging to merge stimulation-response samples across sessions. To address these challenges, we propose MiSO (MicroStimulation Optimization), a closed-loop stimulation framework to drive neural population activity toward specified states by optimizing over a large stimulation parameter space. MiSO consists of three key components: 1) a neural activity alignment method to merge stimulation-response samples across sessions, 2) a statistical model trained on the merged samples to predict the brain's response to untested stimulation parameter configurations, and 3) an online optimization algorithm to adaptively update the stimulation parameter configuration based on the model's predictions. In this study, we implemented MiSO with a factor analysis (FA) based alignment method, a convolutional neural network (CNN), and an epsilon greedy optimization algorithm. We tested MiSO in closed-loop experiments using electrical microstimulation in the prefrontal cortex of a non-human primate. Guided by the CNN predictions, MiSO successfully searched amongst thousands of stimulation parameter configurations to drive the neural population activity toward specified states. More broadly, MiSO increases the clinical viability of neuromodulation technologies by enabling the use of many-fold larger stimulation parameter spaces.

## 1 Introduction

Brain stimulation is an important tool for treating brain disorders [1–3] and for causally perturbing neural activity states to understand brain function [4]. Because complex brain functions are realized through the coordinated activity of populations of neurons, brain stimulation techniques to control neural population activity have the potential to manipulate complex brain function. Most brain stimulation studies to date have used electrical microstimulation to produce a motor response [5] or perceptual experience [6, 7], disrupt neural activity [8, 9], or shift cognitive state [10]. It has been less common to consider the response of a population of neurons to microstimulation [11–14]. In principle, different combinations of stimulation parameters make it possible to create diverse

neural population activity patterns [15–20]. However, it is challenging to identify specific stimulation parameters that achieve a desired activity manipulation for an entire recorded neural population.

Previous studies have developed closed-loop stimulation methods to search the stimulation parameter space [12, 21, 22]. A key challenge for closed-loop stimulation is to widely sample the multi-dimensional stimulation parameter space. It is necessary to collect many stimulation-response samples before the closed-loop stimulation procedure can begin to work effectively. A key innovation of our work is to merge stimulation-response samples across experimental sessions [23–25]. The ability to merge data leverages the finding that neural population activity tends to occupy a lower-dimensional space than the number of neurons being recorded, which enables the alignment of low-dimensional spaces across sessions. By merging data across multiple previous sessions, it is possible for the closed-loop stimulation procedure to work effectively starting from the very outset of the current session.

In this study, we propose MiSO (MicroStimulation Optimization), a closed-loop stimulation framework to drive neural population activity toward specified states by optimizing over a large stimulation parameter space (Section 2.1). MiSO consists of three key components: 1) a neural activity alignment method to merge stimulation-response samples across sessions (Sections 2.2 and 2.3), 2) a statistical model trained on the merged samples to predict the brain's response to untested stimulation parameter configurations (Section 2.4), and 3) a closed-loop optimization algorithm to adaptively update the model's predictions and then choose the next stimulation parameter configuration to test (Section 2.5). In this study, we implemented MiSO with a factor analysis (FA) based alignment method [23], a convolutional neural network (CNN) model [26], and an epsilon greedy optimization algorithm [27]. These methods were chosen to satisfy the fast online computation requirement during the closed-loop stimulation experiments with a predictive model trained on a limited amount of data. However, MiSO is a general framework and applicable to other design choices.

We tested MiSO using electrical microstimulation (uStim) in a non-human primate implanted with a multi-electrode array in the prefrontal cortex (PFC, area 8Ar). MiSO was used to optimize the location of the stimulated electrode(s) on each trial through closed-loop updates, while keeping other parameter values such as current amplitude and frequency fixed. We considered a large uStim parameter space, defined by all possible patterns in which two electrodes out of 96 electrodes are stimulated (4,560 patterns). MiSO's latent alignment method enabled us to merge neural activity across sessions to increase the number of stimulation-response samples (Section 3.1). MiSO's CNN model trained on these merged stimulation-response samples accurately predicted neural responses to untested uStim parameter configurations (Section 3.2). In closed-loop experiments with a non-human primate, MiSO successfully searched among 4,560 uStim parameter configurations to drive the neural population activity toward targeted states. By enabling the search of a larger stimulation parameter space, MiSO produced novel population activity patterns which were not achievable by searching over a smaller stimulation parameter space (Section 3.3).

## 2 Methods

### 2.1 MiSO overview

The goal of MiSO is to identify stimulation parameter configurations that produce specified neural population activity states (Fig. 1A). MiSO first identifies a reference low-d latent space of the high-d population activity to which the neural activity from all the other experimental sessions are aligned (Fig. 1B, Section 2.2). MiSO then collects stimulation-response samples on multiple experimental sessions, then merges the samples using latent space alignment (Section 2.3). The samples are used to fit a statistical model to predict the brain's response to all possible stimulation parameter configurations within the set of parameters defined by the user (Section 2.4). These model predictions obtained prior to the closed-loop experimental session(s) are used to initialize the optimization procedure. At the beginning of each closed-loop experimental session, MiSO identifies the latent space for the session. MiSO then aligns it to the reference latent space, which was used to generate the model predictions (Section 2.5). On each trial, MiSO stimulates using the chosen parameters and updates the predictions based on the responses measured online (Fig. 1C). MiSO iteratively runs this optimization over trials to produce the specified latent activity state.

In the following sections, we refer to a specific stimulation parameter configuration as a "stimulation pattern" and the induced brain response as the "stimulation response". The stimulation patterns

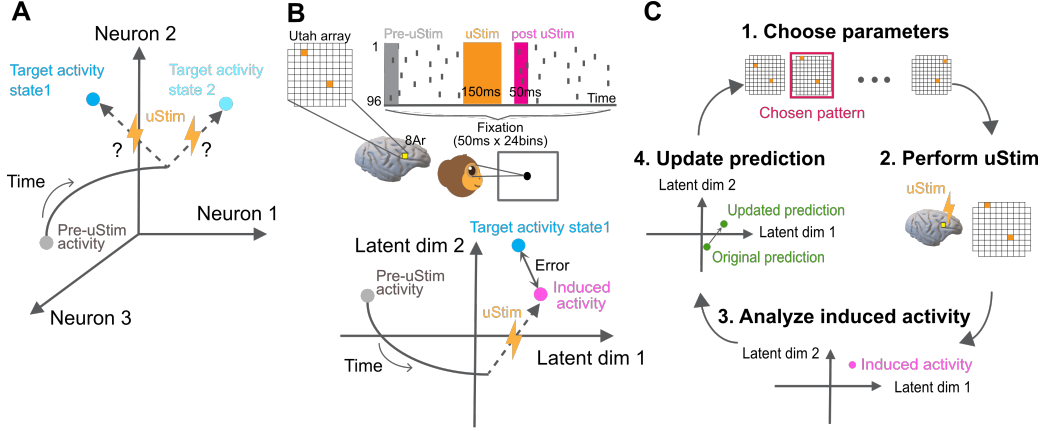

Figure 1: **Experimental paradigm and MiSO closed-loop framework.** (A) MiSO's goal is to optimize brain stimulation parameter configurations to create specified neural population activity states. (B) Experimental setup. Top: Spiking activity was recorded from a multi-electrode array implanted in PFC. During fixation, uStim was applied for 150ms (orange bar) to induce a specified neural population activity state in the post uStim period (pink bar). Bottom: The uStim response was evaluated within a low-d latent space (e.g., 2D) identified from high-d multi-electrode spiking activity. (C) Closed-loop stimulation framework. Each MiSO iteration involves four steps (Section 2.5).

tested in this study consist of all possible patterns in which one of 96 electrodes is stimulated (single electrode stimulation patterns) and/or two of 96 electrodes are stimulated (double electrode stimulation patterns). While the neural activity used in this study consists of spiking responses recorded with a planar grid of electrodes implanted in the brain and each stimulation pattern specifies the location of the electrodes used for stimulation within the grid, MiSO can be readily applied to other stimulation/recording protocols (e.g., holographic optogenetics [19]).

## 2.2 Latent space identification

To merge stimulation-response samples across multiple experimental sessions, MiSO identifies a low-d latent space of the high-d population activity in each session. For this study, we used Factor Analysis (FA), which provides computationally fast latent space identification and latent state estimation, compatible with closed-loop experiments. Since MiSO's ultimate goal is to create targeted changes in neural population activity that lead to changes in brain state and in turn behavior, MiSO identifies the intrinsic latent space only using non-stimulation trials.

For the $i$th session, MiSO extracts a list of $n_i$ "usable" electrodes $\boldsymbol{e_i} \in \mathbb{R}^{n_i}$, where each element of $\boldsymbol{e_i}$ is an integer representing one of the electrodes. An electrode is deemed "usable" if it satisfies criteria involving its mean firing rate, Fano factor, and coincident spiking with other electrodes (see Section S1). For each time bin indexed by $k = 1, ..., K_i^{noStim}$, MiSO then takes spike counts during the fixation period (Fig. 1B) on each usable electrode $\boldsymbol{x}_{i,k}^{noStim} \in \mathbb{R}^{n_i}$. MiSO fits the following FA model using the EM algorithm:

$$\boldsymbol{z}_{i,k}^{noStim} \sim \mathcal{N}(0, I)$$
$$\boldsymbol{x}_{i,k}^{noStim} | \boldsymbol{z}_{i,k}^{noStim} \sim \mathcal{N}(\Lambda_i \boldsymbol{z}_{i,k}^{noStim} + \boldsymbol{\mu_i}, \Psi_i) \tag{1}$$

where $\boldsymbol{z}_{i,k}^{noStim} \in \mathbb{R}^m$, with $m < n_i$, is the low-d latent activity for the $k$th time bin, $\Lambda_i \in \mathbb{R}^{n_i \times m}$ is the loading matrix whose columns define the low-d latent space, $\boldsymbol{\mu_i} \in \mathbb{R}^{n_i}$ contains the mean spike counts for each electrode, and $\Psi_i \in \mathbb{R}^{n_i \times n_i}$ is a diagonal matrix capturing the independent variance of the spike counts for each electrode.

## 2.3 Latent space alignment

To merge neural activity across experimental sessions, we used the latent space alignment method introduced in [23] (termed "FA+Procrustes"). It has been shown to work well in a brain-computer

interface, which like MiSO is a closed-loop experimental paradigm. The FA+Procrustes method solves the Procrustes problem to find an orthogonal transformation matrix to maximize the alignment between two FA loading matrices. First, MiSO runs a single experimental session exclusively with non-stimulation trials and extracts a list of usable electrodes $e_0$, which contains $n_0$ electrodes. MiSO uses the data from this session to identify a reference latent space $\Lambda_0 \in \mathbb{R}^{n_0 \times m}$. For each subsequent session, MiSO aligns the latent space identified in the $i$th session $\Lambda_i$ to the reference latent space $\Lambda_0$. Concretely, for the $i$th session, MiSO identifies an orthogonal transformation matrix $\hat{O}_i \in \mathbb{R}^{m \times m}$ that fulfills:

$$\hat{O}_i = \operatorname*{argmin}_{O:OO^\mathsf{T}=I} \|\Lambda_0(e_{com}, :) - \Lambda_i(e_{com}, :)O^\mathsf{T}\|_F^2 \tag{2}$$

where $e_{com} = e_0 \cap e_i$ is a list of usable electrodes common to the reference session and the $i$th session, and $\|\cdot\|_F$ is the Frobenius Norm. This optimization can be solved in closed-form [28]. The $\hat{O}_i$ found is applied to $\Lambda_i$ to obtain the aligned latent space $\tilde{\Lambda}_i \in \mathbb{R}^{n_i \times m}$:

$$\tilde{\Lambda}_i = \Lambda_i \hat{O}_i^\mathsf{T} \tag{3}$$

The latent activity in the post-stimulation period (Fig. 1B) is estimated as the posterior mean from the FA model (equation (1)) using the loading matrix $\tilde{\Lambda}_i$:

$$z_{i,k}^{Stim} = \beta_i(x_{i,k}^{Stim} - \mu_i) \tag{4}$$

where $z_{i,k}^{Stim} \in \mathbb{R}^m$, $x_{i,k}^{Stim} \in \mathbb{R}^{n_i}$, and $\beta_i = \tilde{\Lambda}_i^\mathsf{T}(\tilde{\Lambda}_i\tilde{\Lambda}_i^\mathsf{T} + \Psi_i)^{-1}$. By using $\tilde{\Lambda}_i$, the induced latent activity $z_{i,k}^{Stim}$ resides in a common latent space across all sessions.

### 2.4 Stimulation-response sample collection and CNN model fitting

To predict the stimulation response to untested stimulation patterns, MiSO uses a statistical model trained on the merged neural activity across sessions. We used a CNN in this study to capture the spatial structure of the stimulation-response relationship across the multi-electrode array (see Sections 2.7 and 3.2 for model selection). To train the CNN, MiSO runs two phases of experimental sessions to collect stimulation-response samples. In the first phase, MiSO applies randomly-selected stimulation patterns to train an initial CNN model. In the second phase, MiSO uses the trained CNN to choose stimulation patterns to maximize the diversity of the observed stimulation responses.

#### 2.4.1 Phase 1: Random stimulation pattern selection

In the first phase, we apply random stimulation patterns over $R$ experimental sessions to train the initial CNN model. With a planar grid of electrodes of size $h \times v$, the stimulation pattern tested in the $k$th trial during the $i$th session $S_{i,k} \in \mathbb{R}^{h \times v}$ is encoded using a value of 1 for stimulated electrodes and 0 for all other electrodes. The induced latent activity state $z_{i,k}^{Stim} \in \mathbb{R}^m$ is computed using equation (4). The entries of $z_{i,k}^{Stim}$ corresponding to the user defined $t$ target dimensions within the $m$ dimensional aligned latent space ($t \leq m$) are subselected and collected in a new vector $\check{z}_{i,k}^{Stim} \in \mathbb{R}^t$. All tested stimulation patterns $S_{i,k}$ and corresponding responses $\check{z}_{i,k}^{Stim}$ across $R$ sessions are appended to obtain $S_k \in \mathbb{R}^{h \times v}$ and $\check{z}_k^{Stim} \in \mathbb{R}^t$, where $k = 1, ..., K^{Stim}$, and $K^{Stim}$ is the total number of stimulation trials across sessions.

The totality of the stimulation-response samples $S_k$ and $\check{z}_k^{Stim}$ are used to train the CNN model, which maps the stimulation patterns $S_k$ to the induced responses $\check{z}_k^{Stim}$. Since the CNN predictions can be variable due to training using a finite number of stimulation-response samples, MiSO uses bagging to stabilize the CNN predictions [29]. In bagging, *M* CNN models are fit with bootstrapping and the top *C* performing models in testing are used to generate predicted simulation responses for all *P* possible stimulation patterns defined by the user (e.g., $P = 4,560$ for double electrode stimulation patterns, computed as "96 choose 2" electrodes) (Section S2). For $p = 1, \ldots, P$, the predicted response $\hat{z}_p \in \mathbb{R}^t$ to the $p$th stimulation pattern $S_p \in \mathbb{R}^{h \times v}$ is defined as the average prediction of the *C* CNN models.

#### 2.4.2 Phase 2: Guided stimulation pattern selection

Because the stimulation responses do not uniformly occupy the $t$-dimensional response space, choosing random stimulation patterns in Phase 1 tends to yield responses primarily in the densest

portions of the response distribution. To diversify the stimulation responses for retraining the CNN, MiSO collects stimulation-response samples using a guided sample collection strategy over $G$ experimental sessions in Phase 2. In each session, MiSO uses the CNN-predicted stimulation responses $\hat{z}_p$ from Phase 1 to select stimulation patterns on the fringes (i.e., sparsely populated regions) of the response distribution. Concretely, for each stimulation pattern $p = 1, ..., P$, MiSO computes the distance measure:

$$d_p = \frac{1}{K} \sum_{j \in N_p} \|\hat{z}_p - \hat{z}_j\|_2^2 \tag{5}$$

where $N_p$ is the set of K-nearest neighbors of $\hat{z}_p$. MiSO chooses the top $E_g$ stimulation patterns that maximize this criterion, corresponding to responses on the fringes of the response distribution. To complement these patterns, MiSO also selects $E_r$ stimulation patterns at random (from the $P$ possible stimulation patterns, typically $E_r < E_g$), corresponding to responses in the more densely populated regions of the response distribution. The selected $E_g + E_r$ patterns are experimentally tested for one session. The resulting stimulation-response samples are appended to the samples collected in Phase 1 and thus far in Phase 2, and the CNN is retrained using the same bagging procedure as in Phase 1. The CNN predictions are then updated and used to choose stimulation patterns to test in the next experimental session. MiSO iterates these steps of selecting stimulation patterns, collecting new stimulation-response samples, and retraining the CNN over $G$ experimental sessions. The CNN predictions $\hat{z}_p$ for $p = 1, ..., P$ obtained during the final session are used to initialize the closed-loop optimization.

## 2.5 Closed-loop optimization

Each online closed-loop session starts with $K_c^{noStim}$ calibration trials, where no stimulation is applied. MiSO extracts a list of usable electrodes $e_c$, which includes $n_c$ electrodes, using these trials and finds the common electrodes $e_{com} = e_c \cap e_0$ between this session and the reference session. The observed spike count vectors $x_{c,k}^{noStim} \in \mathbb{R}^{n_c}$ are used to fit the FA parameters $\Lambda_c \in \mathbb{R}^{n_c \times m}$, $\mu_c \in \mathbb{R}^{n_c}$, and $\Psi_c \in \mathbb{R}^{n_c \times n_c}$. The identified latent space $\Lambda_c$ is aligned to the reference latent space $\Lambda_0$ using the methods described in Section 2.3, yielding $\tilde{\Lambda}_c \in \mathbb{R}^{n_c \times m}$.

The closed-loop optimization starts by the user defining a target state $\check{z}_{targ} \in \mathbb{R}^t$ and loading the predicted stimulation responses $\hat{z}_p$ for $p = 1, ..., P$ obtained from Section 2.4. MiSO then randomly chooses a stimulation pattern $S_s$ from the $P$ possible patterns ($s = 1, \ldots, P$) to start the closed-loop optimization procedure. Subsequent stimulation patterns are chosen using the epsilon greedy algorithm, described below. We used the epsilon greedy algorithm due to the need for a fast online update to compensate for activity fluctuations during the closed-loop optimization.

On the $k$th trial during the closed-loop optimization, the selected stimulation pattern $S_s$ is applied and the induced response during the post-stimulation period $x_{c,k}^{Stim} \in \mathbb{R}^{n_c}$ is measured. MiSO estimates the latent activity $z_{c,k}^{Stim} \in \mathbb{R}^m$ in real time as:

$$z_{c,k}^{Stim} = \beta_c (x_{c,k}^{Stim} - \mu_c) \tag{6}$$

where $\beta_c = \tilde{\Lambda}_c^{\mathsf{T}} (\tilde{\Lambda}_c \tilde{\Lambda}_c^{\mathsf{T}} + \Psi_c)^{-1}$. The entries of $z_{c,k}^{Stim}$ corresponding to the target dimensions are subselected as $\check{z}_{c,k}^{Stim} \in \mathbb{R}^t$, and the prediction of the response to stimulation pattern $S_s$ is updated as:

$$\begin{aligned} \delta &= \hat{z}_s - \check{z}_{c,k}^{Stim} \\ \hat{z}_s &= \hat{z}_s + \alpha_s \delta \end{aligned} \tag{7}$$

where $0 < \alpha_s < 1$ is the learning rate. This update is essential for ensuring that the CNN predictions (based on data from previous experimental sessions) are adjusted for the current closed-loop session. Although the CNN predictions used to initialize the closed-loop optimization can capture spatial structure in the stimulation-response relationship, this update is performed separately for each stimulation pattern and therefore does not leverage spatial structure (see Section 4). MiSO uses a time-varying clipped learning rate:

$$\alpha_s = \max(\alpha_{clip}, \frac{1}{N(s)}) \tag{8}$$

where $0 < \alpha_{clip} < 1$, and $N(s)$ is the number of trials tested with the $s$th stimulation pattern within the current closed-loop session. The clipped learning rate allows MiSO to make larger updates at the beginning of the session, when the framework tries to adapt to stimulation responses in the current session, and smaller updates as the framework becomes more confident about the predictions.

After updating the prediction, MiSO chooses the stimulation pattern to test on the next trial using an epsilon greedy algorithm, which balances exploitation and exploration. With a probability of $1 - \varepsilon$, MiSO chooses the stimulation pattern which minimizes the L1 distance between the induced activity and the target state

$$s^* = \underset{s}{\operatorname{argmin}} \|\hat{\boldsymbol{z}}_s - \check{\boldsymbol{z}}_{targ}\|_1 \tag{9}$$

and applies stimulation pattern $S_{s^*}$ on the next trial. With a probability of $\varepsilon$, MiSO chooses a random stimulation pattern among the $P$ possible patterns. MiSO then iterates until the experimental session is terminated.

## 2.6 Experiment details

We tested MiSO using electrical microstimulation (uStim) in a macaque monkey with a 96 electrode Utah array implanted in prefrontal cortex (PFC, area 8Ar). Experimental procedures were approved by the Institutional Animal Care and Use Committee (IACUC) of Carnegie Mellon University. In each experimental session, the monkey performed a visually-guided saccade task. Each trial began with the monkey fixating on a dot at the center of the screen (Section S3). On uStim trials, we applied uStim for 150 ms during the fixation period. For each session indexed by $i$, we identified latent dimensions by applying FA to spike counts taken in 50 ms bins from usable electrodes $\boldsymbol{e_i}$ that passed a set of criteria (Section S1). Stimulation-response samples were collected starting 50 ms after uStim offset (Fig. 1B, pink) to avoid uStim artifacts.

The number of random stimulation sessions $R$ and guided stimulation sessions $G$ to collect stimulation-response samples for CNN training (Section 2.4) were chosen based on the number of trials the animal worked in each session (about 300-500 uStim trials per session). We performed $R = 3$ sessions with random uStim pattern sampling and $G = 2$ sessions with guided uStim pattern sampling. On each guided sampling session, we collected $E_g = 80$ patterns based on CNN predictions and $E_r = 20$ random patterns. We used the top $C = 10$ test performing models among $M = 50$ CNN models to obtain the CNN prediction. To calculate $d_p$, we used 3 nearest neighbors of $\hat{\boldsymbol{z}}_p$.

Each closed-loop experimental session started with $K_c^{noStim} = 100$ no-uStim trials for latent space calibration. For some sessions, this calibration period also included an additional 96 trials (one per single-electrode stimulation pattern). These observations were used to compare the performance of the "MiSO with single elec., sample avg." with the "Single elec., same day observation" (Fig. 2B). The latter method uses stimulation-response samples collected in the current session to initialize the predictions (Section S4). We chose $t = 2$ target dimensions among the $m = 4$ latent dimensions. The latent dimensionality $m$ was determined based on the cross-validated data likelihood. We set the learning rate $\alpha_{clip}$ as 0.1, chosen manually by assessing how the latent activity state changed over trials, and $\varepsilon$ as 0.05, chosen by running simulations with previously-collected stimulation-response samples.

## 2.7 Model comparison

To decide which prediction model to use in MiSO, we compared the prediction performance of uStim responses when holding out different percentages of uStim patterns in three models: Multi-Layer Perceptron (MLP) [30], Gaussian Process (GP) [31], and Convolutional Neural Network (CNN) [26] (Fig. 3). For all models, we only encoded the location of the stimulating electrodes as inputs in this study. Other stimulation parameters, such as stimulation amplitude and frequency, were constant across electrodes and trials (see Section S3). The models differed in their ability to capture any spatial structure present in the stimulation-response relationship, whereby stimulating with nearby electrodes could induce similar latent activity states.

Below we describe the input (i.e., stimulation pattern) encoding format for each model. For the $k$th trial on the $i$th session, the MLP model takes as input $S_{i,k}^{MLP} \in \{0, 1\}^{96}$, a one-hot vector representing each uStim pattern. Its entries have a value of 1 for each stimulated electrode and 0 for non-stimulated electrodes. Since the one-hot input does not designate which electrodes are close

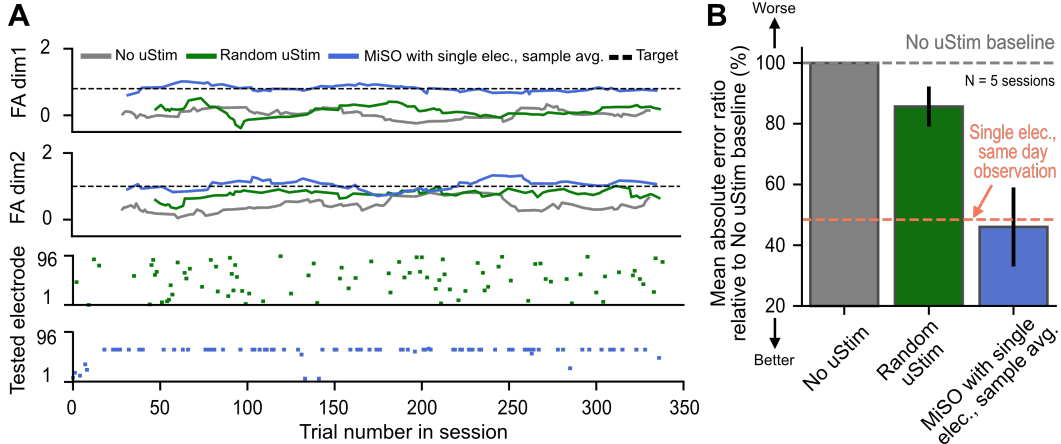

Figure 2: **Closed-loop performance in a non-human primate of MiSO initialized using merged samples.** (A) An example closed-loop experimental session. The top two panels show smoothed (for visualization) FA latent activity in the two target dimensions. Trials for all three methods were interleaved in the session. The bottom two panels show the electrode selected for uStim on each trial by "Random uStim" and "MiSO with single elec., sample avg.". (B) Mean L1 error relative to the "No uStim" baseline across 5 closed-loop experimental sessions. Error bars indicate standard error across sessions.

to or far from each other, this MLP model does not capture spatial structure among the stimulating electrodes. For the $k$th trial on the $i$th session, the GP model takes as input $S_{i,k}^{GP} \in \mathbb{R}^{2n}$, where $n$ is the number of electrodes used for stimulation. The locations of each stimulating electrode are encoded by a pair of values $\{0, 1, ..., 9\} \times \{0, 1, ..., 9\}$, with the stimulating electrodes sorted in ascending order by electrode id. This input encoding format for the GP model captures spatial structure using a single Radial Basis Function kernel. Note that this format requires all stimulation patterns to have the same number of stimulating electrodes. For the $k$th trial on the $i$th session, the CNN takes a grid-format input with the same layout as the electrode array $S_{i,k}^{CNN} \in \{0, 1\}^{10 \times 10}$, where electrodes being used for stimulation have a value of 1 and all other electrodes have a value of 0. The CNN captures spatial structure using multiple 2D convolutional filters applied to this grid input.

## 3 Results

### 3.1 Merging stimulation-response samples across sessions

When the stimulation parameter space is large, the number of uStim patterns we can test within an experimental session is much smaller than the total number of possible uStim patterns. This requires merging neural activity across sessions to create a large enough set of stimulation-response samples to learn their relationship. To develop MiSO, we considered merging neural activity based on aligning latent spaces across sessions [23]. To assess the effectiveness of this method for uStim, we empirically measured the uStim response for all possible single-electrode uStim patterns (96 patterns). We ran 5 experimental sessions in which we stimulated with each of the 96 electrodes on the array for 3-7 trials per session. We found that raw uStim responses were inconsistent across sessions due to neural recording instabilities but were more consistent within the aligned latent space (Fig. S1). This opened up the possibility of estimating uStim responses based on merged stimulation-response samples collected in previous sessions.

To test this idea, we ran closed-loop experiments with all single-electrode uStim patterns in which we used the sample average of the merged stimulation-response samples from previous sessions to initialize the predicted stimulation responses $\hat{z}_p$ in MiSO (Section 2.5). We call this method "MiSO with single elec., sample avg." (Section S4). Here, we could use this method instead of the CNN model to generate the predictions $z_p$, given that for the small parameter space of single electrodes

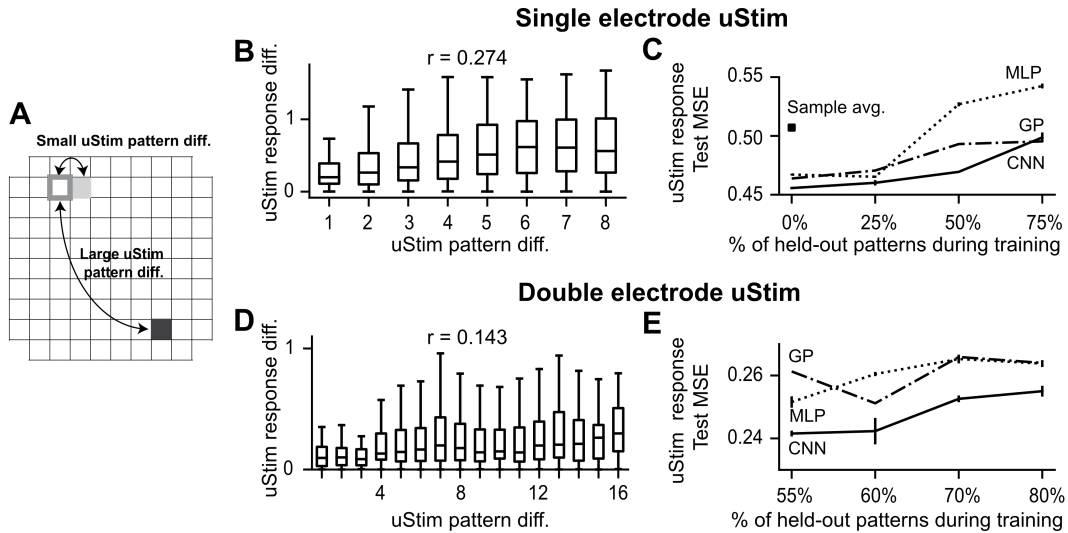

Figure 3: **Leveraging spatial smoothness to predict uStim responses to untested uStim patterns.**
(A) uStim pattern difference determined by the physical location(s) of the stimulating electrode(s)
on the array (illustrated here for single-electrode patterns). (B) Relationship between uStim pattern
difference (horizontal axis, L1 distance, Section S5) and uStim response difference (vertical axis,
latent activity difference along FA dim1) when stimulating using single electrodes. A positive
correlation ($r$) implies that stimulating using nearby electrodes tended to induce similar responses.
(C) uStim response prediction error as a function of the percentage of held-out uStim patterns during
training. Error bars indicate standard error across test datasets. The black square indicates the test
MSE achieved using the sample average of the training data. (D, E) Same format as (B) and (C)
respectively, but for stimulation using double-electrode patterns. In (E), we experimentally tested
45% of all possible double-electrode patterns (9 sessions, 3301 trials).

we could collect sufficient stimulation-response samples for all stimulation patterns to compute the
average responses. We compared this method to two baselines, "No uStim" and "Random uStim"
(Section S4), which allow us to assess whether MiSO induces activity closer to the target state than via
natural activity fluctuations or random uStim patterns, respectively. On each trial, we randomly chose
one of the three methods. "MiSO with single elec., sample avg." (Fig. 2A, blue lines and dots) drove
neural activity closer to the target state than "No uStim" and "Random uStim" (Fig. 2A, gray and
green lines). Across multiple sessions with a different target state on each session, MiSO achieved
significantly smaller errors than the two baseline methods (Fig. 2B, $N = 5$ sessions, $p < 0.05$ for
"No uStim" and "Random uStim", one-tailed t-test). These results indicate that MiSO can identify
effective stimulation patterns by leveraging merged samples from previous sessions.

To further validate the utility of merging past observations, we compared MiSO's performance with a
method where the predictions were initialized using stimulation-response samples (one uStim trial
for each of 96 electrodes) acquired immediately before starting the closed-loop optimization. We call
this method "Single elec., same day observation". This method and MiSO use different approaches to
initialize the same closed-loop procedure (equations (6)-(9)). MiSO leverages stimulation-response
samples collected across multiple previous sessions for initialization. By contrast, "Single elec., same
day observation" uses the stimulation-response samples that can be collected in the current session
for initialization, with the advantage that it gets to observe responses in the current session. We
found that "MiSO with single elec., sample avg." performed equivalently to "Single elec., same day
observation" (Fig. 2B, $N = 5$ sessions, $p = 0.715$, two-tailed t-test). This result indicates that the
merging of stimulation-response samples across previous sessions in MiSO enables the closed-loop
stimulation procedure to work effectively from the very outset of the current session. Furthermore,
this result using single-electrode stimulation patterns represents an important building block for
optimizing over larger stimulation parameter spaces, where merging stimulation-response samples
across sessions becomes essential.

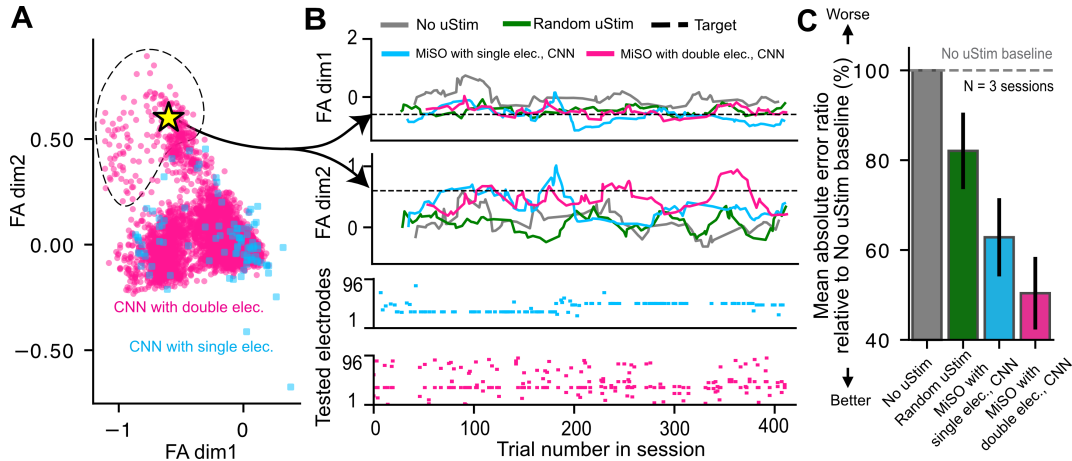

Figure 4: **Closed-loop performance in a non-human primate of MiSO initialized using a CNN.**
(A) Range of activity patterns achievable by single (blue) and double (pink) electrode uStim patterns, as predicted by the CNN. Dashed region: area reachable exclusively by double-electrode uStim. Yellow star: target state used in (B). (B) Example closed-loop experimental session. Same format as Fig. 2A. (C) Mean L1 error of different methods relative to "No uStim" baseline, across 3 closed-loop sessions. Error bars indicate standard error across sessions.

## 3.2 Predicting uStim responses using a CNN

When a search space is small (e.g., 96 single electrodes), we can measure the responses to all uStim patterns to initialize MiSO (as we did with "MiSO with single elec., sample avg.", Section 3.1). This method becomes untenable as we increase size of the stimulation parameter space (e.g., stimulating two electrodes out of 96 yields 4,560 uStim patterns, which would require more than 30 experimental sessions to collect 3 repeats of each uStim pattern). Thus, to expand the stimulation parameter space with MiSO, we need to reduce the required number of stimulation-response samples to learn their relationship.

We asked whether there was spatial structure in the stimulation-response relationship, which could be leveraged to predict the uStim response with a limited number of samples (Fig. 3A). For this, we ran 5 experimental sessions in which we stimulated with each of the 96 electrodes on the array for 3-7 trials per session. With single-electrode uStim, we indeed observed spatial smoothness in the uStim response across physical locations on the array (Fig. 3B, Fig. S2A). In other words, stimulating with nearby electrodes tended to induce similar responses. We then asked whether we could use this property to predict the response to untested uStim patterns. We found that the GP and CNN models more accurately predicted uStim responses than the MLP model when holding out uStim patterns, indicating that spatial smoothness is a useful property for generalization to untested uStim patterns (Fig. 3C).

To increase the size of the stimulation parameter space, we ran 9 experimental sessions in which we stimulated with two out of 96 electrodes (4,560 possible patterns). We were only able to test 45 percent of this large number of possible uStim patterns, with most of the uStim patterns tested for one trial each. As with the single-electrode uStim patterns, we also observed spatial smoothness in the double-electrode uStim patterns (Fig. 3D, Fig. S2B). By leveraging this spatial smoothness, the CNN achieved better generalization performance to untested uStim patterns than the other models (Fig. 3E). Although both the GP and CNN capture aspects of spatial smoothness, the CNN outperformed the GP for double-electrode uStim. Whereas the CNN uses multiple convolution filters to capture the spatial structure of uStim patterns, the GP uses just one spatial kernel, likely limiting its performance. Furthermore, there is no natural input encoding format for a GP when stimulating using more than one electrode on a 2D array. Overall, these results indicate that the CNN model can generalize its predictions to even untested uStim patterns, enabling MiSO to perform closed-loop optimization over a large stimulation parameter space with a limited number of stimulation-response samples.

### 3.3 Closed-loop testing of MiSO initialized using a CNN

By combining the latent space alignment method (Section 3.1) and the CNN prediction model (Section 3.2), we tested MiSO in closed-loop experiments in a non-human primate over a large uStim parameter space, defined by all possible double-electrode uStim patterns (4,560 patterns). To train the CNN model, we ran 2,122 double-electrode uStim trials across 5 sessions (representing ~30 percent of the possible patterns) and merged the stimulation-response samples using latent space alignment. Using the trained CNN model, we predicted the uStim response for all double-electrode uStim patterns. The key motivation of expanding the stimulation parameter space is to expand the range of uStim responses that can be produced. Based on the CNN predictions, we found that the range of activity states achievable by double-electrode stimulation patterns spanned novel regions of the target space that were not reachable using single-electrode uStim (Fig. 4A, Fig. S3).

To demonstrate the benefit of searching a larger stimulus parameter space, we ran closed-loop experiments in which we set the target states to those exclusively reachable using double-electrode uStim patterns (Fig. 4A, yellow star). We implemented "MiSO with double elec., CNN", in which CNN predictions $\hat{z}_p$ were used to initialize the closed-loop optimization (Section 2.5). We compared the performance of "MiSO with double elec., CNN" with three baselines: "No uStim", "Random uStim", as well as MiSO optimizing over the 96 single-electrode uStim patterns (termed "MiSO with single elec., CNN"). If the CNN predictions are meaningful and the closed-loop optimization is able to guide the search in the stimulus parameter space, "MiSO with double elec., CNN" would outperform the three baselines. Indeed, "MiSO with double elec., CNN" (Fig. 4B, pink lines and dots) successfully identified double-electrode uStim patterns that induced responses closer to the target state than the three baseline methods (gray, green, and blue lines). Across multiple sessions with different target states, "MiSO with double elec., CNN" produced smaller errors than the three baseline methods (Fig. 4C, $N = 3$ sessions, $p < 0.05$ for "No uStim" and "Random uStim", and $p = 0.079$ for "MiSO with single elec., CNN", one-tailed t-test). These results demonstrate the utility of MiSO, which enables searching over a larger stimulation parameter space than previously possible.

## 4 Discussion

In this work, we proposed MiSO which searches a large stimulation parameter space to drive neural population activity toward specified states. MiSO's closed-loop optimization procedure is guided by predictions of a CNN trained on merged neural activity across sessions. To our knowledge, this is the first study to apply latent space alignment in the field of brain stimulation to merge stimulation-response samples across sessions. This enables MiSO to search larger stimulation parameter spaces than previously possible.

Although we demonstrated MiSO's functionality in a 2D target space ($t = 2$), the neural activity corresponding to a desired brain state might need to be specified in a higher dimensional space ($t > 2$). As the dimensionality of the target space grows, it may become necessary to expand the stimulation parameter space in order to achieve a greater variety of target states. Provided that a target state is achievable, the epsilon greedy algorithm would require more time to identify appropriate stimulus parameter configurations. The reason is that a stimulation pattern that successfully achieves a target in two dimensions would not necessarily produce the desired activity if a third dimension was added to the target specification. In this case, the use of a more efficient online algorithm would be beneficial to reduce the exploration of suboptimal stimulation parameter configurations.

To expand the range of achievable population activity states, one might consider expanding the stimulation parameter space. For example, in the context of electrical microstimulation, we could use stimulation patterns involving larger numbers of electrodes and/or different current amplitudes or frequencies. For the CNN model (or any other predictive model) to retain its prediction accuracy, it would likely need to capture additional structure in the stimulation-response relationship (e.g., similar stimulation amplitudes or frequencies lead to similar responses) and require more training data. Furthermore, MiSO's closed-loop updates of the predicted responses are performed separately for each stimulation pattern (equation (7)), which becomes untenable as the stimulation parameter space grows. In this case, incorporating a statistical model in the closed-loop updates to account for spatial (or other) structure could be beneficial, such that observing a response to one stimulation pattern leads to updates of the predicted responses for multiple stimulation patterns. These computations would need to be performed fast enough to be used in a closed-loop optimization procedure.

## 5 Acknowledgments

This work was supported by NIH CRCNS R01 MH118929, NSF NCS DRL 2124066, Simons Foundation 543065 and NC-GB-CULM-00003241-05, and Japan Student Services Organization fellowship. We are grateful to Samantha Schmitt for assistance with data collection, and Karen McCracken and Mary Ellen Smyth for animal care.

## References

[1] Helen S Mayberg, Andres M Lozano, Valerie Voon, Heather E McNeely, David Seminowicz, Clement Hamani, Jason M Schwalb, and Sidney H Kennedy. Deep brain stimulation for treatment-resistant depression. *Neuron*, 45(5):651–660, 2005.

[2] Robert S Fisher and Ana Luisa Velasco. Electrical brain stimulation for epilepsy. *Nature Reviews Neurology*, 10(5):261–270, 2014.

[3] Andres M Lozano, Nir Lipsman, Hagai Bergman, Peter Brown, Stephan Chabardes, Jin Woo Chang, Keith Matthews, Cameron C McIntyre, Thomas E Schlaepfer, Michael Schulder, et al. Deep brain stimulation: current challenges and future directions. *Nature Reviews Neurology*, 15(3):148–160, 2019.

[4] Marlene R Cohen and William T Newsome. What electrical microstimulation has revealed about the neural basis of cognition. *Current opinion in neurobiology*, 14(2):169–177, 2004.

[5] Charles J Bruce, Michael E Goldberg, M Catherine Bushnell, and Gregory B Stanton. Primate frontal eye fields. ii. physiological and anatomical correlates of electrically evoked eye movements. *Journal of neurophysiology*, 54(3):714–734, 1985.

[6] Giles S Brindley and Walpole S Lewin. The sensations produced by electrical stimulation of the visual cortex. *The Journal of physiology*, 196(2):479–493, 1968.

[7] John S Choi, Austin J Brockmeier, David B McNiel, Lee M Von Kraus, José C Príncipe, and Joseph T Francis. Eliciting naturalistic cortical responses with a sensory prosthesis via optimized microstimulation. *Journal of neural engineering*, 13(5):056007, 2016.

[8] Mark M Churchland and Krishna V Shenoy. Delay of movement caused by disruption of cortical preparatory activity. *Journal of neurophysiology*, 97(1):348–359, 2007.

[9] Nuo Li, Kayvon Daie, Karel Svoboda, and Shaul Druckmann. Robust neuronal dynamics in premotor cortex during motor planning. *Nature*, 532(7600):459–464, 2016.

[10] Tirin Moore and Mazyar Fallah. Microstimulation of the frontal eye field and its effects on covert spatial attention. *Journal of neurophysiology*, 91(1):152–162, 2004.

[11] Mark H Histed, Vincent Bonin, and R Clay Reid. Direct activation of sparse, distributed populations of cortical neurons by electrical microstimulation. *Neuron*, 63(4):508–522, 2009.

[12] Sina Tafazoli, Camden J MacDowell, Zongda Che, Katherine C Letai, Cynthia R Steinhardt, and Timothy J Buschman. Learning to control the brain through adaptive closed-loop patterned stimulation. *Journal of Neural Engineering*, 17(5):056007, 2020.

[13] Yuxiao Yang, Shaoyu Qiao, Omid G Sani, J Isaac Sedillo, Breonna Ferrentino, Bijan Pesaran, and Maryam M Shanechi. Modelling and prediction of the dynamic responses of large-scale brain networks during direct electrical stimulation. *Nature biomedical engineering*, 5(4):324–345, 2021.

[14] Amin Nejatbakhsh, Francesco Fumarola, Saleh Esteki, Taro Toyoizumi, Roozbeh Kiani, and Luca Mazzucato. Predicting the effect of micro-stimulation on macaque prefrontal activity based on spontaneous circuit dynamics. *Physical Review Research*, 5(4):043211, 2023.

[15] Chieko M Murasugi, C Daniel Salzman, and William T Newsome. Microstimulation in visual area MT: effects of varying pulse amplitude and frequency. *Journal of Neuroscience*, 13(4):1719–1729, 1993.

[16] Alexis M Kuncel and Warren M Grill. Selection of stimulus parameters for deep brain stimulation. *Clinical neurophysiology*, 115(11):2431–2441, 2004.

[17] Meghan Watson, Numa Dancause, and Mohamad Sawan. Intracortical microstimulation parameters dictate the amplitude and latency of evoked responses. *Brain Stimulation*, 9(2):276–284, 2016.

[18] Thierri Callier, Nathan W Brantly, Attilio Caravelli, and Sliman J Bensmaia. The frequency of cortical microstimulation shapes artificial touch. *Proceedings of the National Academy of Sciences*, 117(2):1191–1200, 2020.

[19] Hillel Adesnik and Lamiae Abdeladim. Probing neural codes with two-photon holographic optogenetics. *Nature neuroscience*, 24(10):1356–1366, 2021.

[20] Marcus Triplett, Marta Gajowa, Hillel Adesnik, and Liam Paninski. Bayesian target optimisation for high-precision holographic optogenetics. *Advances in Neural Information Processing Systems*, 36, 2024.

[21] Samuel Laferriere, Marco Bonizzato, Sandrine L Côté, Numa Dancause, and Guillaume Lajoie. Hierarchical bayesian optimization of spatiotemporal neurostimulations for targeted motor outputs. *IEEE Transactions on Neural Systems and Rehabilitation Engineering*, 28(6):1452–1460, 2020.

[22] Yuri B Saalmann, Sima Mofakham, Charles B Mikell, and Petar M Djuric. Microscale multicircuit brain stimulation: Achieving real-time brain state control for novel applications. *Current Research in Neurobiology*, 4:100071, 2023.

[23] Alan D Degenhart, William E Bishop, Emily R Oby, Elizabeth C Tyler-Kabara, Steven M Chase, Aaron P Batista, and Byron M Yu. Stabilization of a brain–computer interface via the alignment of low-dimensional spaces of neural activity. *Nature biomedical engineering*, 4(7):672–685, 2020.

[24] Brianna M Karpowicz, Yahia H Ali, Lahiru N Wimalasena, Andrew R Sedler, Mohammad Reza Keshtkaran, Kevin Bodkin, Xuan Ma, Lee E Miller, and Chethan Pandarinath. Stabilizing brain-computer interfaces through alignment of latent dynamics. *BioRxiv*, pages 2022–04, 2022.

[25] Max Dabagia, Konrad P Kording, and Eva L Dyer. Aligning latent representations of neural activity. *Nature Biomedical Engineering*, 7(4):337–343, 2023.

[26] Yann LeCun, Yoshua Bengio, and Geoffrey Hinton. Deep learning. *Nature*, 521(7553):436–444, 2015.

[27] Richard S Sutton. Reinforcement learning: An introduction. *A Bradford Book*, 2018.

[28] Peter H Schönemann. A generalized solution of the orthogonal procrustes problem. *Psychometrika*, 31(1):1–10, 1966.

[29] Leo Breiman. Bagging predictors. *Machine learning*, 24:123–140, 1996.

[30] David E Rumelhart, Geoffrey E Hinton, and Ronald J Williams. Learning representations by back-propagating errors. *Nature*, 323(6088):533–536, 1986.

[31] Christopher KI Williams and Carl Edward Rasmussen. *Gaussian processes for machine learning*, volume 2. MIT press Cambridge, MA, 2006.

[32] Jacob R Gardner, Geoff Pleiss, David Bindel, Kilian Q Weinberger, and Andrew Gordon Wilson. Gpytorch: Blackbox matrix-matrix gaussian process inference with gpu acceleration. In *Advances in Neural Information Processing Systems*, 2018.

# Supplementary material

## S1 Spiking activity preprocessing

To identify the latent dimensions using FA, we computed binned spike counts during the 1.2 s fixation period on each no-uStim trial (50 ms bins, yielding 24 bins per trial). Thus, $K_i^{noStim}$ for the $i$th session was the number of no-uStim trials $\times$ 24. These no-uStim trials were also used to extract a list of usable electrodes $\boldsymbol{e_i}$ for the $i$th session based on the following three criteria: mean firing rate >1 Hz, Fano factor <8, and <20% coincident spiking with each of the other electrodes. The latent dimensionality ($m = 4$) was chosen based on the cross-validated data likelihood across multiple sessions, then used for all sessions. To evaluate the uStim response, we computed binned spike counts starting 50 ms after uStim offset on each uStim trial (Fig. 1B) to avoid uStim artifacts. The latent activity was then computed using equation (4).

## S2 Models architecture and fitting

The CNN architecture and hyperparameters were determined based on a grid search. The following describes the CNN architecture used in this work from input to output: 1. a convolutional layer with 32 channels, spatial kernel size 3x3, and stride size 1, followed by ReLU activation, 2. a convolutional layer with 64 channels, spatial kernel size 3x3, and stride size 1, followed by ReLU activation, 3. two linear layers with ReLU activation, and 4. a linear output layer of size equal to the target dimensionality without any activation. The MLP architecture used in this work includes: 1. an input layer of size 96, 2. a hidden layer of size 10, followed by ReLU activation, and 3. an output layer of size equal to the target dimensionality. The CNN and MLP were implemented in PyTorch and fit using the Adam optimizer with mean squared error loss and a learning rate of 0.001. The GP model was fitted using the GPyTorch library [32]. We used a Radial Basis Function whose length scale hyperparameter was chosen by maximizing the marginal data log likelihood using the Adam optimizer with a learning rate of 0.001. We trained all models on a local computing cluster using 4 NVIDIA GeForce RTX GPUs and 11GB of RAM. We used the same architecture and hyperparameters for all experiments.

To train CNN models using bagging (Section 2.4), the merged stimulation-response samples across sessions were split into training, validation, and test sets with a ratio of 80:10:10. Each CNN model was trained on the training data. The validation data was used for hyperparameter tuning. The test data were used to evaluate the ability of the models to generalize to untested uStim patterns, and to choose the top $C$ performing models to generate the uStim response predictions for all possible uStim patterns. These predictions were used to initialize the closed-loop optimization (Section 2.5).

## S3 Details of uStim experimental paradigm

In each experimental session, the monkey performed a visually-guided saccade task. In this task, the monkey first fixated on a dot at the center of the screen. Following a random (1.45-1.75s) fixation period, the center dot turned off and one of four peripheral targets (45°, 135°, 225°, 315°) appeared. The monkey saccaded to that target to receive a liquid reward. There were two types of visually-guided saccade trials: "uStim trials", in which we applied uStim, and "no-uStim trials", in which we did not apply uStim. The experimental system randomly chose which trial type to perform in an interleaved manner. On uStim trials, we applied uStim for 150 ms during the fixation period. The stimulation was biphasic with each square pulse in the biphasic pair being 250 ms in duration. We set the current amplitude low enough not to induce any eye movements (current amplitude: 25 uA for single-electrode uStim, 15 uA + 15 uA for double-electrode uStim). We changed the location(s) of the stimulated electrode(s) on each trial, while keeping other parameter values such as current amplitude and frequency (350 Hz) fixed. In each closed-loop session, we chose two target dimensions along which we could modulate neural activity and induce diverse latent activity with uStim. These were not necessarily the top FA dimensions that explained the greatest covariance among the neurons.

## S4 Closed-loop performance assessment methods

The table below summarizes the methods used to assess closed-loop performance. The "Prediction model" column indicates the method used to obtain uStim response predictions to initialize the closed-loop optimization. The "Training data" column indicates the stimulation-response samples used to train the prediction model. The "uStim patterns" column indicates the number of electrodes used for stimulation, both in the training data and during closed-loop optimization. The "Online algorithm" column indicates the method used to choose the next uStim pattern. All methods that use the epsilon greedy algorithm also perform closed-loop updates of the predicted responses (equations (6)-(9)).

| Method | Pred. model | Training data | uStim patterns | Online algorithm |
|---|---|---|---|---|
| No uStim | None | None | None | None |
| Random uStim | None | None | Single elec. for Section 3.1<br>Double elec. for Section 3.3 | Random selection |
| Single elec., same day observation | Sample avg. | Same day | Single elec. | Epsilon greedy |
| MiSO with single elec., sample avg. | Sample avg. | Multi day | Single elec. | Epsilon greedy |
| MiSO with single elec., CNN | CNN | Multi day | Single elec. | Epsilon greedy |
| MiSO with double elec., CNN | CNN | Multi day | Double elec. | Epsilon greedy |

## S5 Spatial structure of uStim responses

To evaluate the spatial smoothness of uStim responses across the multi-electrode array (Fig. 3), we measured the spatial similarity between uStim patterns ($d_{pattern}$) and the similarity in the uStim response elicited by those patterns ($d_{response}$). We computed these metrics for every pair of tested uStim patterns.

For single-electrode uStim patterns $e_1 = [x_1, y_1]$ and $e_2 = [x_2, y_2]$ (for $x_i$ and $y_i \in \{0, 1, ..., 9\}$), we quantified their spatial similarity using the L1 distance:

$$d_{pattern} = |x_1 - x_2| + |y_1 - y_2| \tag{10}$$

where $x_1$ and $y_1$ are the spatial coordinates on the multi-electrode grid for the first uStim electrode $e_1$, and $x_2$ and $y_2$ are the spatial coordinates for the second uStim electrode $e_2$.

For double-electrode uStim patterns, each pattern involves two electrodes. Let $e_1 = [x_1, y_1]$ and $e_2 = [x_2, y_2]$ represent the uStim electrodes used for the first pattern, and $e_3 = [x_3, y_3]$ and $e_4 = [x_4, y_4]$ represent the uStim electrodes used for the second pattern. To calculate the distance in this case, we computed:

$$\begin{aligned}
d_{pattern}^1 &= (|x_1 - x_3| + |y_1 - y_3|) + (|x_2 - x_4| + |y_2 - y_4|) \\
d_{pattern}^2 &= (|x_1 - x_4| + |y_1 - y_4|) + (|x_2 - x_3| + |y_2 - y_3|)
\end{aligned} \tag{11}$$

and used $\min(d_{pattern}^1, d_{pattern}^2)$ as the distance measure.

For each pair of uStim patterns, we quantified the difference in uStim response using the L1 distance:

$$d_{response} = |z_1 - z_2| \tag{12}$$

where $z_1$ is the response to the first uStim pattern along a target dimension, and $z_2$ is the uStim response to the second uStim pattern along the same target dimension. To focus on local smoothness, we only analyzed the pairs in which $d_{pattern}$ was less than 8 in the single-electrode case, and less than 16 in the double-electrode case.

Using the $d_{pattern}$ and $d_{response}$ values, we computed the correlation between them to evaluate if spatial structure is present. A positive correlation indicates that uStim electrodes located closer to each other on the array tend to induce a more similar response.

## S6 Summary of experimental sessions

The experimental sessions used in this study comprised 3 sessions without uStim (reference sessions), 10 sessions involving single-electrode uStim, 19 sessions involving double-electrode uStim, and 8 closed-loop sessions. Each of the entries in the tables below uses a subset of the sessions described above.

The table below summarizes the number of sessions used in each closed-loop experimental result:

|  | Reference session | Training sessions | Closed-loop test sessions |
|---|---|---|---|
| Fig. 2 | 1 session | 5 sessions | 5 sessions |
| Fig. 4 | 1 session | 5 sessions for single elec. CNN<br>5 sessions for double elec. CNN | 3 sessions |

The table below summarizes the number of sessions used in each offline analysis result:

|  | Reference session | Training sessions | Test sessions |
|---|---|---|---|
| Fig. 3B, Fig. 3C, and Fig. S1A | 1 session | 5 sessions | 5 sessions |
| Fig. 3D, Fig. 3E, and Fig. S1B | 1 session | 9 sessions | 5 sessions |
| Fig. S3 | 1 session | 10 sessions (split into train and test) | |

## S7 Code availability

Python code for MiSO is available on GitHub at https://github.com/yuumii-san/MiSO.git.

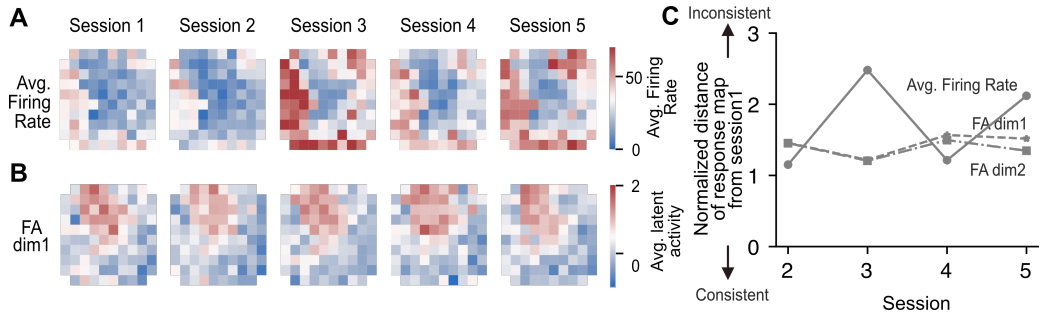

Figure S1: **uStim response consistency across sessions.** (A) Response maps of uStim-induced mean firing rates. Each panel shows the mean firing rate, averaged across the entire array, induced by stimulating each electrode individually. For example, the color of the cell in (x,y) position (1,2) in a given response map indicates the mean firing rate across the array induced by stimulating this particular electrode. Each column corresponds to a different session. (B) Response maps of uStim-induced mean latent activity across trials. Each panel shows the average latent activity induced by stimulating each electrode individually. The latent spaces have been aligned across sessions. Stimulation-response samples from the five sessions shown here are used to compute the sample average-based predictions in Fig. 2. (C) Normalized distance of response maps from session 1. The mean firing rates and latent activity are normalized across sessions using a min-max transformation to align their scales. The uStim response is more consistent across sessions in the aligned latent space than in the raw firing rate space.

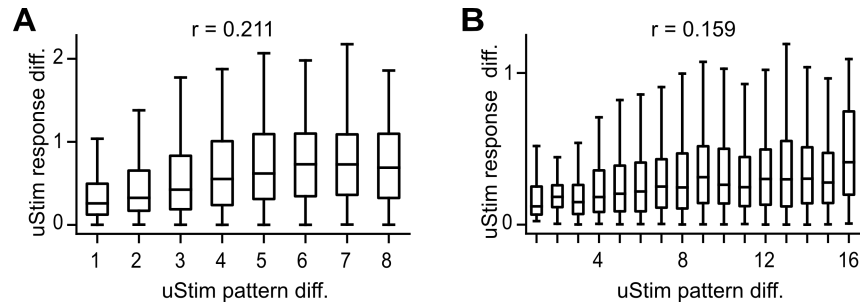

Figure S2: **Relationship between uStim pattern spatial similarity and response similarity along the second target dimension.** Same format as Fig. 3B and Fig. 3D, which were based on the first target dimension. Left, single-electrode uStim, right, double-electrode uStim.

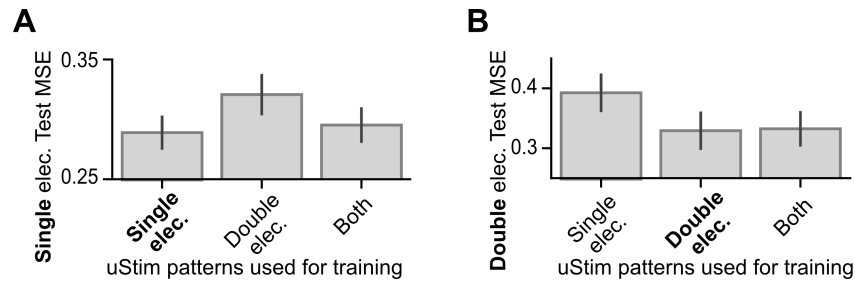

Figure S3: **Generalization to uStim patterns involving a different number of electrodes can be challenging.** (A) Prediction error for different CNN models tested on single-electrode uStim data. Each bar shows the predictive performance of CNNs trained with different merged stimulation-response samples (single-electrode uStim trials, double-electrode uStim trials, and both types of trials). A CNN trained on single-electrode uStim trials performed better than a CNN trained on double-electrode uStim trials. Because the single-electrode training trials include all 96 possible patterns, we are not testing here the ability of the CNN to generalize to unseen uStim patterns (see Fig. 3). Instead, we are assessing to what extent the CNN is able to predict the response to double-electrode uStim patterns given only training data from single-electrode uStim patterns. If the responses to double-electrode uStim were simply a linear combination of the responses to single-electrode uStim using each constituent electrode, then the performance of the CNN trained on double-electrode uStim trials should perform as well as the CNN trained on single-electrode uStim trials. However, this was not the case. These results reveal a complex relationship between single-electrode uStim responses and double-electrode uStim responses. Furthermore, training a CNN on both types of uStim trials yielded similar accuracy to training only on single-electrode uStim trials. (B) Prediction error for different CNN models tested on double-electrode uStim data. Here the double-electrode uStim trials used to train the CNN comprised 1,031 (of the possible 4,560) uStim patterns, which were partially overlapping with the double-electrode uStim patterns used for testing. Even though this requires some generalization to double-electrode uStim patterns not used in training, the CNN trained on double-electrode uStim trials still outperformed the CNN trained on single-electrode uStim trials. This provides further evidence for a complex relationship between single-electrode uStim responses and double-electrode uStim responses.

